# Theory and Dynamics of Perceptual Bistability

**Paul R. Schrater**[*]
Departments of Psychology and Computer Sci. & Eng.
University of Minnesota
Minneapolis, MN 55455
schrater@umn.edu

**Rashmi Sundareswara**
Department of Computer Sci. & Eng.
University of Minnesota
sundares@cs.umn.edu

## Abstract

Perceptual Bistability refers to the phenomenon of spontaneously switching between two or more interpretations of an image under continuous viewing. Although switching behavior is increasingly well characterized, the origins remain elusive. We propose that perceptual switching naturally arises from the brain's search for best interpretations while performing Bayesian inference. In particular, we propose that the brain explores a posterior distribution over image interpretations at a rapid time scale via a sampling-like process and updates its interpretation when a sampled interpretation is better than the discounted value of its current interpretation. We formalize the theory, explicitly derive switching rate distributions and discuss qualitative properties of the theory including the effect of changes in the posterior distribution on switching rates. Finally, predictions of the theory are shown to be consistent with measured changes in human switching dynamics to Necker cube stimuli induced by context.

## 1 Introduction

Our visual system is remarkably good at producing consistent, crisp percepts of the world around us, in the process hiding interpretation uncertainty. Perceptual bistability is one of the few circumstances where ambiguity in the visual processing is exposed to conscious awareness. Spontaneous switching of perceptual states frequently occurs during continuously viewing an ambiguous image, and when a new interpretation of a previously stable stimuli is revealed (as in the sax/girl in figure **??**a), spontaneous switching begins to occur[**?**]. Moreover, although perceptual switching can be modulated by conscious effort[**?**, **?**], it cannot be completely controlled.

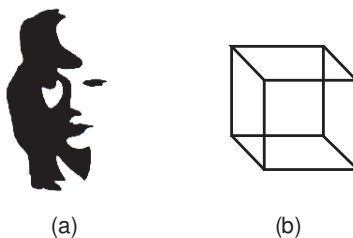

(a)　　　(b)

Figure 1: Examples of ambiguous figures: (a) can be interpreted as a woman's face or a saxophone player. (b) can be interpreted as a cube viewed from two different viewpoints.

Stimuli that produce bistability are characterized by having several distinct interpretations that are in some sense equally plausible. Given the successes of Bayesian inference as a model of perception

---

[*]http://www.schrater.org

(for instance [**?**, **?**, **?**]), these observations suggest that bistability is intimately connected with making perceptual decisions in the presence of a multi-modal posterior distribution, as previously noted by several authors[**?**, **?**]. However, typical Bayesian models of perceptual inference have no dynamics, and probabilistic inference per se provides no reason for spontaneous switching, raising the possibility that switching stems from idiosyncrasies in the brain's implementation of probabilistic inference, rather than from general principles.

In fact, most explanations of bistability have been historically rooted in proposals about the nature of neural processing of visual stimuli, involving low-level visual processes like retinal adaptation and neural fatigue[**?**, **?**, **?**]. However, the abundance of behavioral and brain imaging data that show high level influences on switching (like intentional control which can produce 3-fold changes in alternation rate[**?**]) have revised current views toward neural hypotheses involving combinations of both sensory and higher order cortical processing[**?**]. The goal of this paper is to provide a simple explanation for the origins of bistability based on general principles that can potentially handle both top-down and bottom-up effects.

## 2  Basic theory

The basic ideas that constitute our theory are simple and partly form standard assumptions about perceptual processing. The core assumptions are:

1. Perception performs Bayesian inference by exploring and updating the posterior distribution across time by a kind of sampling process (e.g. [**?**]).
2. Conscious percepts result from a decision process that picks the interpretations by finding sample interpretations with the highest posterior probability (possibly weighted by the cost of making errors).
3. The results of these decisions and their associated posterior probabilities are stored in memory until a better interpretation is sampled.
4. The posterior probability associated with the interpretation in memory decays with time.

The intuition behind the model is that most percepts of objects in a scene are built up across a series of fixations. When an object previously fixated is eccentrically viewed or occluded, the brain should store the previous interpretation in memory until better data comes along or the memory becomes too old to be trusted. Finally, the interpretation space required for direct Bayesian inference is too large for even simple images, but sampling schemes may provide a simple way to perform approximate inference.

The theory provides a natural interface to interpret both high-level and low-level effects on bistability, because any event that has an impact on the relative heights or positions of the modes in the posterior can potentially influence durations. For example, patterns of eye fixations have long been known to influence the dominant percept[**?**]. Because eye movement events create sudden changes in image information, it is natural that they should be associated with changes in the dominant mode. Similarly, control of information via selective attention and changes in decision thresholds offer concrete loci for intentional effects on bistability.

## 3  Analysis

To analyze the proposed theory, we need to develop temporal distributions for the maxima of a multi-modal posterior based on a sampling process and describe circumstances under which a current sample will produce an interpretation better than the one in memory. We proceed as follows. First we develop a general approximation to multi-modal posterior distributions that can vary over time, and analyze the probability that a sample from the posterior are close to maximal. We then describe how the samples close to the max interact with a sample in memory with decay.

A tractable approximation to a multi-modal distribution can be formed using a mixture of uni-modal distributions centered at each maxima.

$$P(\theta_t|D_{0:t}) = \frac{P(D_t|\theta_t)P(\theta_t|D_{0:t-\Delta t})}{P(D_t|D_{0:t-\Delta t})} \approx \sum_{i=1}^{\#maxima} p_i(D_t|\theta_t; \theta_{t,i}^*)P_i(\theta|D_{0:t-\Delta t}; \theta_{t,i}^*) \quad (1)$$

where $\theta_t$ is the vector of unknown parameters (e.g. shape for Necker Cube) at time $t$, $\theta_{t,i}^*$ is the location of the maxima of the $i^{th}$ mode, $D_t$ is the most recent data, $D_{0:t-\Delta t}$ is the data history, and $P_i(\theta|D_{0:t-\Delta t};\theta_{t,i}^*)$ is the predictive distribution (prior) for the current data based on recent experience[1].

Near the maxima, the negative log of the uni-modal distributions can be expanded into a second-order Taylor series:

$$-L_i(\theta_t|D_t) \approx d_i^2 + k_i \tag{2}$$
$$= (\theta_t - \theta_{t,i}^*)^T \mathcal{I}_i(\theta_{t,i}^*|D_{0:t})(\theta - \theta_{t,i}^*) + 1/2\log(|\mathcal{I}_i^{-1}|) + c_i \tag{3}$$

where $\mathcal{I}_i(\theta_{t,i}^*|D_{0:t}) = \frac{\partial^2 \log(P(\theta_t|D_{0:t}))}{\partial\theta\partial\theta^T}|_{\theta_{t,i}^*}$ is the observed information matrix and $c_i = \log\left(P(\theta_{t,i}^*|D_{0:t})\right)$ represents the effect of the predictive prior on the posterior height at the $i^{th}$ mode. Thus, samples from a posterior mode will be approximately $\chi^2$ distributed near the maximum with effective degrees of freedom $n$ given by the number of significant eigenvalues of $\mathcal{I}_i^{-1}$. Essentially $n$ encodes the effective degrees of freedom in interpretation space. [2]

## 3.1 Distribution of transition times

We assume that the perceptual interpretation is selected by a decision process that updates the interpretation in memory $m_\theta(t)$ whenever the posterior probability of the most recent sample both exceeds a decision threshold and the discounted probability of the sample in memory. Given these assumptions, we can approximate the probability distribution for update events. Assuming the sampling forms a locally stationary process $d_i^2(t)$, update events involving entry into mode $i$ are first passage times $T_i$ of $d_i^2(t)$ below both the minimum of the current memory sample $\omega_t$ and the decision threshold $\xi$:

$$T_i(\xi,\omega_t) = \min\{t : \delta_t^i \leq \min\{\xi,\omega_t\}\}$$

where $\delta_t^i = d_i^2(t) - k_i$, time $t$ is the duration since the last update event and $\omega_t = \log(P(m_\theta(t)|D_{0:t}))$ is the log posterior of the sample in memory at time $t$. Let $M_t^i = \inf_{0\leq s\leq t}\delta_s^i$.

The probability of waiting at least $t$ for an update event is related to the minima of the process by:
$P(T_i(\xi,\omega_t) < t) = P(M_t^i < \min\{\xi,\omega_t\})$

This probability can be expressed as:

$$P(M_t^i < \min\{\xi,\omega_t\}) = \tag{4}$$
$$\int_0^t p(\delta_\tau^i < \omega_t) \quad p(\omega_t < \xi)P(i|\tau)d\tau + \int_0^t p(\delta_\tau^i < \xi)\left(1 - P(\omega_t < \xi)\right)P(i|\tau)d\tau$$

where $P(i|t) = P(\theta_t \in S_i)$ denotes the probability that a sample drawn between times 0 and $t$ is in the support $S_i$ of the $i^{th}$ mode. To generate tractable expressions from equation ??, we make the following assumptions.

**Memory distribution** Assume that the memory decay process is slow relative to the sampling events, and that the decay process can be modeled as a random walk in the interpretation space $m_\theta(t) = m_\theta(0) + \sum_{0\leq\tau_i\leq t}\epsilon_\theta(\tau_i)$, where $\tau_i$ are sample times, and $\epsilon_\theta$ are small disturbances with zero mean and variance $\sigma$ we assume to be small. Because variances add, the average effect on the distance $\omega_t$ is a linear increase: $\omega_t = \omega_0 + \rho\sigma t$, where $\rho$ is the sampling rate. These disturbances could represent changes in the local of the maxima of the posterior due to the incorporation of new data, neural noise, or even discounting (note that linearly increasing $\omega_t$ corresponds to exponential or multiplicative discounting in probability).

To understand the behavior of this memory process, notice that every $m_\theta(0)$ must be within distance $\xi$ of the maximum of the posterior for an update to occur. Due to the properties of extrema of distributions of $\chi^2$ random variables, an $m_\theta(0)$ will be (in expectation) a characteristic distance $\mu_m(\xi)$ below $\xi$ and for $t > 0$ drifts with linear dynamics[3]. This suggests the approximation, $p(\omega_t < \xi) \approx \delta(\mu_m + \rho\sigma t - \omega_t)$, which can be formally justified because $p(\omega_t < \xi)$ will be highly peaked with respect to the distribution of the sampling process $p(\delta_\tau^i)$. Finally assuming slow drift means $(1 - P(\omega_t < \xi)) \approx 0$ on the time scale that transitions occur[4]. Under these assumptions, equation ?? reduces to:

$$P(M_t^i < \min\{\xi, \omega_t\}) = \int_0^t p(\delta_\tau^i < \omega_t)\delta(\mu_m + \rho\sigma t - \omega_t)P(i|\tau)d\tau \tag{5}$$

$$\approx P(M_t^i < \mu_m + \rho\sigma t)P(i) \tag{6}$$

where $P(i)$ is the average frequency of sampling from the $i^{th}$ mode.

**Extrema of the posterior sampling process**   If the sampling process has no long-range temporal dependence, then under mild assumptions the distribution of extrema converge in distribution[5] to one of three characteristic forms that depend only on the domain of the random variable[?]. For $\chi^2$ samples, the distribution of minima converges to

$$P(M_t^i \leq b) = 1 - \exp(-cNb^{a-1})$$

where $N$ is the number of samples, $c(n) = \frac{2^{-a+1}}{\Gamma(a)}$, $a(n) = \left(\frac{n}{2}\right)$. Set $N = \rho t$ and let $\rho = 1$ for convenience, where $\rho$ is the effective sampling rate, and equation ?? can be written as:

$$P(T_i < t) = P(M_t^i \leq \min\{\xi, \omega_t\}) \approx P(M_t^i < \mu_m + \sigma t)P(i) \tag{7}$$

$$= \left(1 - \exp\left(-c\,t(\mu_m + \sigma t)^{a-1}\right)\right)P(i) \tag{8}$$

The probability distribution shows a range of behavior depending on the values of $a = n/2$ and $\mu_m(\xi)$. Note that the time scale for switching. In particular, for $n > 4$ and $\mu_m(\xi)$ relatively small, the distribution has a gamma-like behavior, where new memory update transitions are suppressed near recent transitions. For $n = 2$, or for $\mu_m(\xi)$ large, the above equation reduces to exponential. This behavior shows the effect of the decision threshold, as without a decision threshold the asymptotic behavior of simple sampling schemes will generate approximately exponentially distributed update event times, as a consequence of extreme value theory. Finally, for $n = 1$and small $\mu_m(\xi)$, the distribution becomes Cauchy-like with extremely long tails. See figure ?? for example distributions. Note that the time scale of events can be arbitrarily controlled by appropriately selecting $\rho$ (controls the time scale of the sampling process) and $\sigma$ (controls the time scale of the memory decay process).

**Effects of posterior parameters on update events**   The memory update distributions are effected primarily by two factors, the log posterior heights and their difference $\Delta k_{ij} = k_i - k_j$, and the effective number of degrees of freedom per mode $n$.

**Effect of $k_i$, $\Delta k_{ij}$**   The variable $\Delta k_{ij}$ has possible effects both on the probability that a mode is sampled, and the temporal distributions. When the modes are strongly peaked (and the sampling procedure is unbiased) $\log P(i) \approx \Delta k_{ij}$. Secondly, $\Delta k_{ij}$ effectively sets different thresholds for each mode, because memory update events occur when:

$$\delta_t^i = d_i^2(t) - k_i > \min\{\omega_t, \xi\}$$

Increasing the effective threshold for mode $i$ makes updates of type $i$ more frequent, and should drive the temporal dynamics of the dominant mode toward exponential. Finally, if the posterior becomes more peaked while the threshold remains fixed, the update rates should increase and the temporal distributions will move toward exponential. If we assume increased viewing time makes the posterior more peaked, then our model predicts the common finding of increased transition rates with viewing duration.

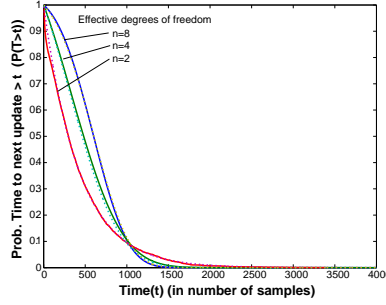

Figure 2: Examples of cumulative distribution functions of memory update times. Solid curves are generated by simulating the sampling with decision process described in the text. Dashed lines represent theoretical curves based on the approximation in equation **??**, showing the quality of the approximation.

**Effect of** $n$   One of the surprising aspects of the theory above is the strong dependence on the effective number of degrees of freedom. The theory makes a strong prediction that stimuli that have more interpretation degrees of freedom will have longer durations between transitions, which appears to be qualitatively true across both rivalry and bistability experiments[**?**, **?**] (of course, depending how you interpret the number of degrees of freedom).

**Relating theory to behavioral data via induced Semi-Markov Renewal Process**   Assuming that memory update events involving transitions to the same mode are not perceptually accessible, only update events that switch modes are potentially measurable. However, the process described above fits the description of a generator for a semi-Markov renewal process. A semi-Markov renewal process involves Markov transitions between discrete states $i$ and $j$ determined by a matrix with entries $\mathbf{P_{ij}}$, coupled with random durations spent in that state sampled from time distributions $F_{ij}(t)$. The product of these distributions $Q_{ij}(t) = \mathbf{P_{ij}}F_{ij}(t)$ is the generator of the process, that describes the conditional probability of first transition between states $\mathbf{i} \to \mathbf{j}$ in time less than $t$, given first entry into state $i$ occurs at time $t = 0$. In the theory above, $F_{ij}(t) = F_{ii}(t) = P(T_i < t)$, while $\mathbf{P_{ij}} = \mathbf{P_{jj}} = P(j)$[6]

The main reason for introducing the notion of a renewal process is that they can be used to express the relationship between the theoretical distributions and observable quantities. The most commonly collected data are times between transitions and (possibly contingent) percept frequencies. Here we present results found in Ross[**?**]. Let the state $s(t) = i$ refer to when the memory process is in the support of mode $i$: $m_\theta(t) \in S_i$ at time $t$. The distribution of first transition times from state $s = i$ can be expressed formally as a cumulative probability of first transition:

$$G_{ij}(t) = P(N_j(t) > 0|s(0) = i) = P(\mathcal{T}_j < t|s(0) = i)$$

where $N_j(t)$ is the number of transitions into state $j$ in $time <= t$, $\mathcal{T}_j$ is the time until first memory update of type $j$. For two state processes, only $G_{01}(t)$ and $G_{10}(t)$ are measurable. Let $P(0)$, denote the probability of sampling from mode 0. The relationship between the generating process and the distribution of first transitions is given by:

$$G_{01}(t) = \int_0^t G_{01}(t - \tau)dQ_{00}(\tau) + Q_{01}(t) \tag{9}$$

$$G_{01}(t) = P(0)\int_0^t G_{01}(t - \tau)\frac{dP(T_0 < \tau)}{dt}d\tau + P(1)P(T_0 < t) \tag{10}$$

which appears only to be solvable numerically for the general form of our memory update transition functions, however, for the case in which $P(T_0 < t)$ is exponential, $G_{01}(t)$ is as well. Moreover,

for gamma-like distributions, the convolution integral tends to increase the shape parameter, which means that gamma parameter estimates produced by fitting transition durations will overestimate the amount of 'memory' in the process[7]. Finally note the limiting behavior as $P(0) \to 0$, $G_{01}(t) = P(T_0 < t)$, so that direct measurement of the temporal distributions is possible *but only for the (almost) supressed perceptual state*. Similar relationships exist for survival probabilities, defined as $S_{ij}(t) = P(s(t) = j|s(0) = i)$

## 4 Experiments

In this section we investigate simple qualitative predictions of the theory, that biasing perception toward one of the interpretations will produce a coupled set of changes in both percept frequencies and durations, under the assumption that perceptual biases result from differences in posterior heights . To bias perception of a bistable stimuli, we had observers view a Necker cube flanked with 'fields of cubes' that are perceptually unambiguous and match one of the two percepts (see figure **??**). Subjects are typically biased toward seeing the Necker cube in the "looking down" state (65-70% response rates), and the context stimuli shown in figure **??**a) have little effect on Necker cube reversals. We found that the looking up context, boosts "looking up" response rates from 30% to 55%.

### 4.1 Methods

Subject's perceptual state while viewing the stimuli in fig. **??** were collected using the methods described in[**?**]. Eye movement effects[**?**] were controlled by having observers focus on a tiny sphere in the center of the Necker cube, and attention was controlled using catch trials. Base rates for reversals were established for each observer (18 total) in a training phase. Each observer viewed 100 randomly generated context stimuli and each stimulus was viewed long enough to acquire 10 responses (taking 10-12 sec on average). For ease of notation, we represent the "Looking down" condition as state $0$ and the "Looking Up" as state $1$.

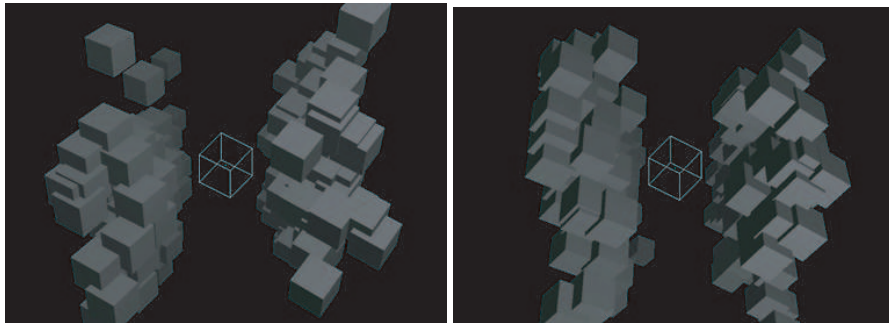

(a) An instance of the "Looking down" con-
text with the Necker cube in the middle

(b) An instance of the "Looking up" context
with the Necker cube in the middle

Figure 3: The two figures are examples of the "Looking down" and "Looking up" context conditions.

### 4.2 Results

We measured the effect of context on estimates of perceptual switching rates, $R_i = P(s(t) = i)$, first transition durations $G_{ij}$, and survival probabilities $P_{ii} = P(s(t) = i|s(0) = i)$ by counting the number of events of each type. Additionally, we fit a semi-Markov renewal process $Q_{ij}(t) = \mathbf{P_{ij}}F_{ij}(t)$ to the data using a sampling based procedure. The procedure is too complex to fully describe in this paper, so a brief description follows. For ease of sampling, $F_{ij}(t)$ were gamma with separate parameters for each of the four conditionals $\{00, 01, 10, 11\}$, resulting in 10 parameters

overall. The process was fit by iteratively choosing parameter values for $Q_{ij}(t)$, simulating response data and measuring the mismatch between the simulated and human $G_{ij}$ and $P_{ii}$ distributions.

The effect of context on $G_{ij}$ and $P_{ii}$ is shown in Fig.**??** and Fig.**??** for the contexts "Looking Down" and "Looking Up" respectively. The figures also show the maximum likelihood fitted gamma functions. Testable predictions generated by simulating the memory process described above were verified, including changes in mean durations of about 2sec, coupling of the duration distributions, and an increase in the underlying renewal process shape parameters when the percepts are closer to equally probable.

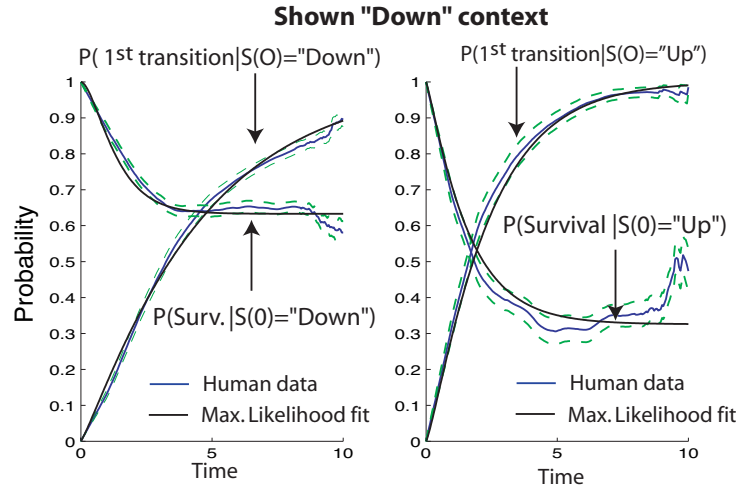

Figure 4: Data pooled across subjects for the "Looking Down" context condition. (a) Prob. of first transition and the survival probability of the "Looking down" percept. (b)Prob. of first transition and conditional survival probability of the "Looking Up" percept. A semi-Markov renewal process with transition paramters $P_{ij}$, gamma means $m_{ij}$ and gamma variances $v_{ij}$ was fit to all the data via max. likelihood. The best fit curves are superimposed on the data.

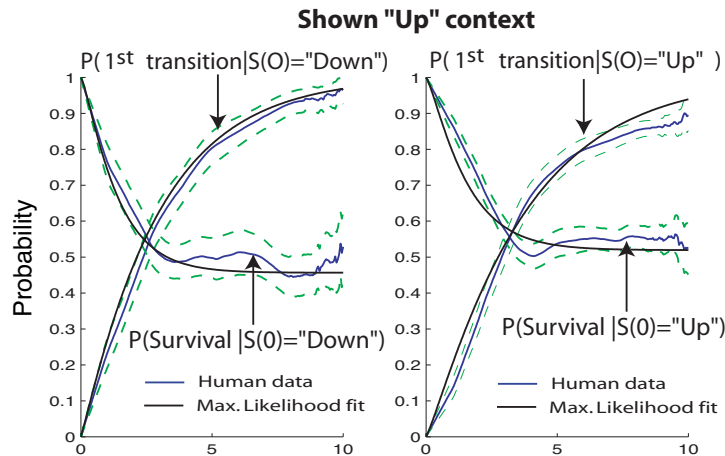

Figure 5: Same as figure **??**, but for the "Looking Up" context condition.

## 5  Discussion/Conclusions

Although [**?**] also presents a theory for average transitions times in bistability based on random processes and a multi-modal posterior distribution, their theory is fundamentally different as it derives

switching events from tunneling probabilities that arise from input noise. Moreover, their theory predicts increasing transition times as the posterior becomes increasingly peaked, exactly opposite our predictions.

In conclusion, we have presented a novel theory of perceptual bistability based on simple assumptions about how the brain makes perceptual decisions. In addition, results from a simple experiment show that manipulations which change the dominance of a percept produce coupled changes in the probability of transition events as predicted by theory. However, we do not regard the experiment as a strong test of the theory. We believe the strength of the theory is that it can make a large set of qualitative predictions about the distribution of transition events by coupling transition times to simple properties of the posterior distribution. Our theory suggests that the basic descriptive model sufficient to capture perceptual bistability is a semi-Markov renewal process, which we showed could successfully simulate the temporal dynamics of human data for the Necker cube.

## Footnotes

[1]Because time is critical to our arguments, we assume that the posterior is updated across time (and hence new data) using a process that resembles Bayesian updating.

[2]For the Necker cube, the interpretation space can be thought of as the depths of the vertices. A strong prior assumption that world angles between vertices are close to 90deg produces two dominate modes in the posterior that correspond to the typical interpretations. Within a mode, the brain must still decide whether the vertices conform exactly to a cube. Thus for the Necker cube, $n$ might be as high as 8 (one depth value per vertex) or as low as 1 (all vertices fixed once the front corner depth is determined).

[3]In the simulations, $\mu_m$ is chosen as the expected value of the set of events below the threshold $xi$

[4]Conversely fast drift in the limit means $P(\omega_t < \xi) \approx 0$, which results in transitions entirely determined by the minima of the sampling process and $\xi$.

[5]Corresponds to the limit assertion $\sup_b |P(M_t \geq b) - \exp(-\lambda(b,t)t)| \to 0$ as $t \to \infty$

[6]The independence relations are a consequence of an assumption of independence in the sampling procedure, and relaxing that assumption can produce state contingencies in $Q_{ij}(t)$. Therefore, we do not consider this to be a prediction of the theory. For example, mild temporal dependence (e.g. MCMC-like sampling with large steps) can create contingencies in the frequency of sampling from the $i^{th}$ mode that will produce a non-independent transition matrix $\mathbf{P_{ij}} = P(\theta_t \in S_i|\theta_{t-\rho\Delta t} \in S_j)$.

[7]gamma shape parameters are frequently interpreted as the number of events in some abstract Poisson process that must occur before transition

# References

[1] Aldous, D .(1989) Probability approximations via the Poisson clumping heuristic. Applied Math. Sci, 77. Springer-Verlag, New York.

[2] Bialek, W., DeWeese, M. (1995) Random Switching and Optimal Processing in the Perception of Ambiguous Signals. *Physics Review Letters* 74(15) 3077-80.

[3] Brascamp, J. W., van Ee, R., Pestman, W. R., & van den Berg, A. V. (2005). Distributions of alternation rates in various forms of bistable perception. *J. of Vision* 5(4), 287-298.

[4] Einhauser, W., Martin, K. A., & Konig, P. (2004). Are switches in perception of the Necker cube related to eye position? *Eur J Neuroscience* 20(10), 2811-2818.

[5] Freeman, W.T (1994) The generic viewpoint assumption in a framework for visual perception *Nature* vol. 368, April 1994.

[6] von Grunau, M. W., Wiggin, S. & Reed, M. (1984). The local character of perspective organization. *Perception and Psychophysics* 35(4), 319-324.

[7] Kersten, D., Mamassian, P. & Yuille, A. (2004) Object Perception as Bayesian Inference *Annual Review of Psychology* Vol. 55, 271-304.

[8] Lee, T.S. & Mumford, D. (2003) Hierarchical Bayesian Inference in the Visual Cortex *Journal of the Optical Society of America* Vol. 20, No. 7.

[9] Leopold, D. and Logothetis, N.(1999) Multistable phenomena: Changing views in Perception. *Trends in Cognitive Sciences*. Vol.3, No.7, 254-264.

[10] Long, G., Toppino, T. & Mondin, G. (1992) Prime Time: Fatigue and set effects in the perception of reversible figures. *Perception and Psychophysics* Vol.52, No.6, 609-616.

[11] Mamassian, P. & Goutcher, R. (2005) Temporal dynamics in Bistable Perception. *Journal of Vision*. No. 5, 361-375.

[12] Rock, I. and Mitchener, K.(1992) Further evidence of the failure of reversal of ambiguous figures by uninformed subjects. *Perception* 21, 39-45.

[13] Ross, S. M. (1970) Applied Probability Models with Optimization Applications. Holden-Day.

[14] Stocker, A. & Simoncelli, E. (2006) Noise characteristics and prior expectations in human visual speed perception *Nature Neuroscience* vol.9, no.4, 578-585.

[15] Toppino, T. C. (2003). Reversible-figure perception: mechanisms of intentional control. *Perception and Psychophysics* 65(8), 1285-1295.

[16] Toppino, T. C. & Long, G. M. (1987). Selective adaptation with reversible figures: don't change that channel. *Perception and Psychophysics* 42(1), 37-48.

[17] van Ee, R., Adams, W. J., & Mamassian, P. (2003). Bayesian modeling of cue interaction: Bi-stability in stereo-scopic slant perception. *J.of the Opt. Soc. of Am. A*, 20, 1398-1406.

[18] van Ee, R., van Dam, L.C.J., Brouwer,G.J. (2005) Dynamics of perceptual bi-stability for stereoscopic slant rivalry. *Vision Res.*, 45, 29-40.
